# Managing Uncertainty in Cue Combination

**Zhiyong Yang**
Department of Neurobiology, Box 3209
Duke University Medical Center
Durham, NC 27710
zhyyang@duke.edu

**Richard S. Zemel**
Department of Psychology
University of Arizona
Tucson, AZ 85721
zemel@u.arizona.edu

## Abstract

We develop a hierarchical generative model to study cue combination. The model maps a global shape parameter to local cue-specific parameters, which in turn generate an intensity image. Inferring shape from images is achieved by inverting this model. Inference produces a probability distribution at each level; using distributions rather than a single value of underlying variables at each stage preserves information about the validity of each local cue for the given image. This allows the model, unlike standard combination models, to adaptively weight each cue based on general cue reliability and specific image context. We describe the results of a cue combination psychophysics experiment we conducted that allows a direct comparison with the model. The model provides a good fit to our data and a natural account for some interesting aspects of cue combination.

Understanding cue combination is a fundamental step in developing computational models of visual perception, because many aspects of perception naturally involve multiple cues, such as binocular stereo, motion, texture, and shading. It is often formulated as a problem of inferring or estimating some relevant parameter, e.g., depth, shape, position, by combining estimates from individual cues.

An important finding of psychophysical studies of cue combination is that cues vary in the degree to which they are used in different visual environments. Weights assigned to estimates derived from a particular cue seem to reflect its estimated reliability in the current scene and viewing conditions. For example, motion and stereo are weighted approximately equally at near distances, but motion is weighted more at far distances, presumably due to distance limits on binocular disparity.[3] Experiments have also found these weightings sensitive to image manipulations; if a cue is weakened, such as by adding noise, then the uncontaminated cue is utilized more in making depth judgments.[9] A recent study[2] has shown that observers can adjust the weighting they assign to a cue based on its relative utility for a particular task. From these and other experiments, we can identify two types of information that determine relative cue weightings: (1) *cue reliability*: its relative utility in the context of the task and general viewing conditions; and (2) *region informativeness*: cue information available locally in a given image.

A central question in computational models of cue combination then concerns how these forms of uncertainty can be combined. We propose a hierarchical generative

model. Generative models have a rich history in cue combination, as they underlie models of Bayesian perception that have been developed in this area.[10,5] The novelty in the generative model proposed here lies in its hierarchical nature and use of distributions throughout, which allows for both context-dependent and image-specific uncertainty to be combined in a principled manner.

Our aims in this paper are dual: to develop a combination model that incorporates cue reliability and region informativeness (estimated across and within images), and to use this model to account for data and provide predictions for psychophysical experiments. Another motivation for the approach here stems from our recent probabilistic framework,[11] which posits that every step of processing entails the representation of an entire probability distribution, rather than just a single value of the relevant underlying variable(s). Here we use separate local probability distributions for each cue estimated directly from an image. Combination then entails transforming representations and integrating distributions across both space and cues, taking across- and within-image uncertainty into account.

# 1   IMAGE GENERATION

In this paper we study the case of combining shading and texture. Standard shape-from-shading models exclude texture,[1,8] while standard shape-from-texture models exclude shading.[7] Experimental results and computational arguments have supported a strong interaction between these cues,[10] but no model accounting for this interaction has yet been worked out.

The shape used in our experiments is a simple surface:

$$Z = B(1 - x^2), |x| <= 1, |y| <= 1 \tag{1}$$

where $Z$ is the height from the $xy$ plane. $B$ is the only shape parameter.

Our image formation model is a hierarchical generative model (see Figure 1). The top layer contains the global parameter $B$. The second layer contains local shading and texture parameters $\mathbf{S}, \mathbf{T} = \{S_i, T_i\}$, where $i$ indexes image regions. The generation of local cues from a global parameter is intended to allow local uncertainties to be introduced separately into the cues. This models specific conditions in realistic images, such as shading uncertainty due to shadows or specularities, and texture uncertainty when prior assumptions such as isotropicity are violated.[4] Here we introduce uncertainty by adding independent local noise to the underlying shape parameter; this manipulation is less realistic but easier to control.

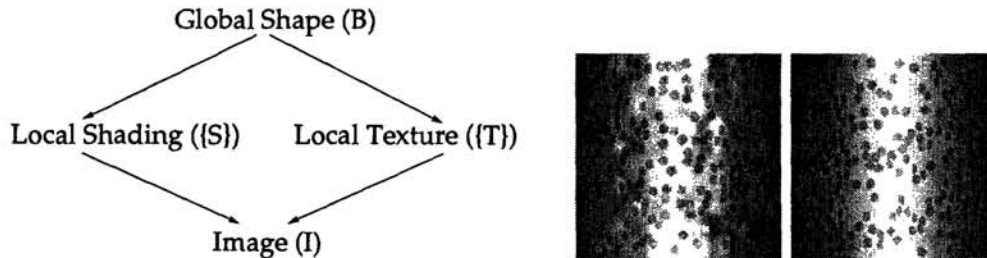

Figure 1: Left: The generative model of image formation. Right: Two sample images generated by the image formation procedure. $B = 1.4$ in both. Left: $\sigma_s = 0.05$, $\sigma_t = 0$. Right: $\sigma_s = 0$, $\sigma_t = 0.05$.

The local cues are sampled from Gaussian distributions: $p(S_i|B) = \mathcal{N}(f(B); \sigma_s)$; $p(T_i|B) = \mathcal{N}(g(B); \sigma_t)$. $f(B), g(B)$ describe how the local cue parameters depend

on the shape parameter $B$, while $\sigma_s$ and $\sigma_t$ represent the degree of noise in each cue. In this paper, to simplify the generation process we set $f(B) = g(B) = B$. From $\{S_i\}$ and $\{T_i\}$, two surfaces are generated; these are essentially two separate noisy local versions of $B$. The intensity image combines these surfaces. A set of same-intensity texsels sampled from a uniform distribution are mapped onto the texture surface, and then projected onto the image plane under orthogonal projection. The intensity of surface pixels not contained within these texsels are determined generated from the shading surface using Lambertian shading. Each image is composed of $10 \times 10$ non-overlapping regions, and contains $400 \times 400$ pixels. Figure 1 shows two images generated by this procedure.

## 2   COMBINATION MODEL

We create a combination, or recognition model by inverting the generative model of Figure 1 to infer the shape parameter $B$ from the image. An important aspect of the combination model is the use of distributions to represent parameter estimates at each stage. This preserves uncertainty information at each level, and allows it to play a role in subsequent inference.

The overall goal of combination is to infer an estimate of $B$ given some image $I$. We derive our main inference equation using a Bayesian integration over distributions:

$$P(B|I) = \int P(B|\mathbf{S}, \mathbf{T})P(\mathbf{S}, \mathbf{T}|I)d\mathbf{S}d\mathbf{T} \tag{2}$$

$$P(\mathbf{S}, \mathbf{T}|I) \sim \prod_i P(S_i|I)P(T_i|I) \tag{3}$$

$$P(B|\mathbf{S}, \mathbf{T}) = P(B)P(\mathbf{S}, \mathbf{T}|B)/\int P(B)P(\mathbf{S}, \mathbf{T}|B)db \sim \prod_i P(S_i|B)P(T_i|B) \tag{4}$$

To simplify the two components we have assumed that the prior over $B$ is uniform, and that the $\mathbf{S}, \mathbf{T}$ are conditionally independent given $B$, and given the image. This third assumption is dubious but is not essential in the model, as discussed below. We now consider these two components in turn.

### 2.1   Obtaining local cue-specific representations from an image

One component in the inference equation, $P(\mathbf{S}, \mathbf{T}|I)$, describes local cue-dependent information in the particular image $I$. We first define intermediate representations $S, T$ that are dependent on shading and texture cues, respectively. The shading representation is the curvature of a horizontal section: $S = f(B) = 2B(1 + 4x^2B^2)^{-3/2}$. The texture representation is the cosine of the surface slant: $T = g(B) = (1 + 4x^2B^2)^{-1/2}$. Note that these $S, T$ variables do not match those used in the generative model; ideally we could have used these cue-dependent variables, but generating images from them proved difficult.

Some image pre-processing must take place in order to estimate values and uncertainties for these particular local variables. The approach we adopt involves a simple statistical matching procedure, similar to $k$-nearest neighbors, applied to local image patches. After applying Gaussian smoothing and band-pass filtering to the image, two representations of each patch are obtained using separate shading and texture filters. For shading, image patches are represented by forming a histogram of $\frac{\Delta I}{I}$; for texture, the patch is represented by the mean and standard deviation of the amplitude of Gabor filter responses at 4 scales and orientations. This representation of a shading patch is then compared to a database of similar

patch representations. Entries in the shading database are formed by first select-ing a particular value of $B$ and $\sigma_s$, generating an image patch, and applying the appropriate filters. Thus $S = f(B)$ and the noise level $\sigma_s$ are known for each entry, allowing an estimate of these variables for the new patch to be formed as a linear combination of the entries with similar representations. An analogous procedure, utilizing a separate database, allows $T$ and an uncertainty estimate to be derived for texture. Both databases have 60 different $b, \sigma$ pairs, and 10 samples of each pair.

Based on this procedure we obtain for each image patch mean values $M_i^s, M_i^t$ and uncertainty values $V_i^s, V_i^t$ for $S_i, T_i$. These determine $P(I|S), P(I|T)$, which are approximated as Gaussians. Taking into account the Gaussian priors for $S_i, T_i$,

$$P(S_i|I) = P(I|S_i)P(S_i) \sim \exp(-\frac{V_i^s}{2}(S-M_i^s)^2)\exp(-\frac{V_0^s}{2}(S-M_0^s)^2) \quad (5)$$

$$P(T_i|I) = P(I|T_i)P(T_i) \sim \exp(-\frac{V_i^t}{2}(T-M_i^t)^2)\exp(-\frac{V_0^t}{2}(T-M_0^t)^2) \quad (6)$$

Note that the independence assumption of Equation 3 is not necessary, as the matching procedure could use a single database indexed by both the shading and texture representations of a patch.

## 2.2   Transforming and combining cue-specific local representations

The other component of the inference equation describes the relationship between the intermediate, cue-specific representations $S, T$ and the shape parameter $B$:

$$P(S|B) \sim \exp(-\frac{V_b^s}{2}(S-f(B))^2) \quad ; \quad P(T|B) \sim \exp(-\frac{V_b^t}{2}(T-g(B))^2) \quad (7)$$

The two parameters $V_b^s, V_b^t$ in this equation describe the uncertainty in the relation-ship between the intermediate parameters $S, T$ and $B$; they are invariant across space. These two, along with the parameters of the priors—$M_0^s, M_0^t, V_0^s, V_0^t$—are the free parameters of this model. Note that this combination model neatly ac-counts for both types of cue validity we identified: the variance in $P(S|B)$ de-scribes the *general uncertainty* of a given cue, while the local variance in $P(S|I)$ describes the *image-specific uncertainty* of the cue.

Combining Equations 3-7, and completing the integral in Equation 2, we have:

$$P(B|I) \sim \exp\left[-\frac{1}{2}\sum_i a_1 f(B)^2 + a_2 g(B)^2 - 2a_3 f(B) - 2a_4 g(B)\right] \quad (8)$$

$a_1 = \frac{V_b^s(V_i^s+V_0^s)}{V_b^s+V_i^s+V_0^s}, a_2 = \frac{V_b^t(V_i^t+V_0^t)}{V_b^t+V_i^t+V_0^t}, a_3 = \frac{V_b^s(V_i^s M_i^s+V_0^s M_0^s)}{V_b^s+V_i^s+V_0^s}, a_4 = \frac{V_b^t(V_i^t M_i^t+V_0^t M_0^t)}{V_b^t+V_i^t+V_0^t}$. Ap-proximating $P(B|I)$ as a Gaussian, we obtain the mean $\mathcal{U}$ and std. deviation $\Sigma$:

$$\frac{\partial \log(-P(B|I))}{\partial B}|_\mathcal{U} = 0 \quad ; \quad \Sigma = [\frac{\partial^2(-\log P(B|I))}{\partial^2 B}|_\mathcal{U}]^{-1/2} \quad (9)$$

Thus our model infers from any image a mean $\mathcal{U}$ and variance $\Sigma^2$ for $B$ as non-linear combinations of the cue estimates, taking into account the various forms of uncertainty.

## 3   A CUE COMBINATION PSYCHOPHYSICS EXPERIMENT

We have conducted psychophysical experiments using stimuli generated by the procedure described above. In each experimental trial, a stimulus image and four

views of a mesh surface are displayed side-by-side on a computer screen. The subject's task is to manipulate the curvature of the mesh to match the stimulus. The final shape of the mesh surface describes the subject's estimate of the shape parameter $B$ on that trial. The subject's variance is computed across repeated trials with an identical stimulus. In a given block of trials, the stimulus may contain only shading information (no texture elements), only texture information uniform shading), or both. The local cue noise $(\sigma_s, \sigma_t)$ is zero in some blocks, non-zero in others. The primary experimental findings (see Figure 2) are:

- Shape from shading alone produces underestimates of $B$. Shape from texture alone also leads to underestimation, but to a lesser degree.

- Shape from both cues leads to almost perfect estimation, with smaller variance than shape from either cue alone. Thus *cue enhancement*—more accurate and robust judgements for stimuli containing multiple cues than just individual cues—applies to this paradigm.

- The variance of a subject's estimation increases with $B$.

- Noise in either shading or texture systematically biases the estimation from the true values: the greater the noise level, the greater the bias.

- Shape from both cues is more robust against noise than shape from either cue alone, providing evidence of another form of cue enhancement.

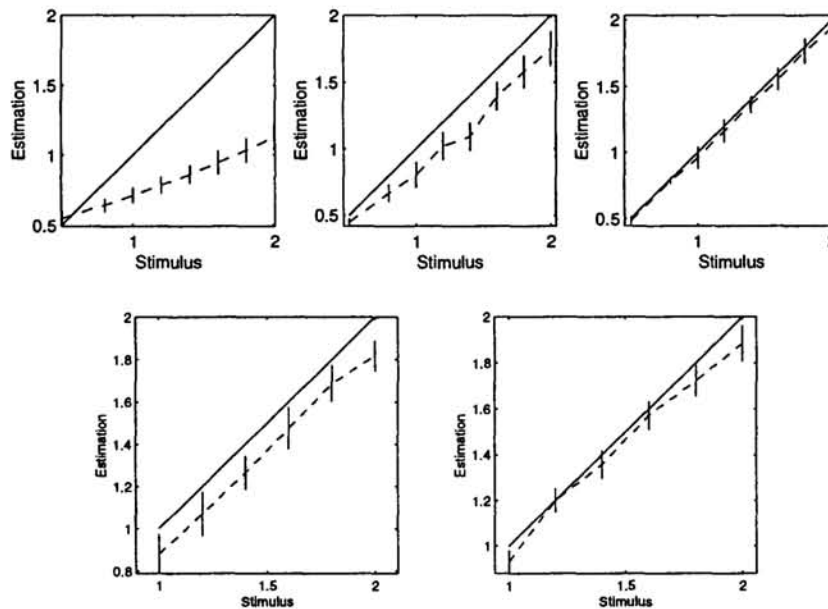

Figure 2: Means and standard errors are shown for the shape matching experiment, for different values of $B$, under different stimulus conditions. TOP: No noise in local shape parameters. Left: Shape from shading alone. Middle: Shape from texture alone. Right: Shape from shading and texture. BOTTOM: Shape from shading and texture. Left: $\sigma_s = 0.05, \sigma_t = 0$. Right: $\sigma_s = 0, \sigma_t = 0.05$.

## 4   MODELING RESULTS

The model was trained using a subset of data from these experiments. The error criteria was mean relative error ($MRE$) between the model outputs ($\mathcal{U}, \Sigma$) and

| $B$ | $\sigma_s$ | $\sigma_t$ | data $(\mathcal{U}/\Sigma)$ | model $(\mathcal{U}/\Sigma)$ |
|------|------|------|-----------------|-----------------|
| 1.4 | 0.10 | 0 | 1.18/0.072 | 1.20/0.06 |
| 1.6 | 0.10 | 0 | 1.34/0.075 | 1.35/0.063 |
| 1.4 | 0.05 | 0 | 1.32/0.042 | 1.4/0.067 |
| 1.6 | 0.05 | 0 | 1.52/0.049 | 1.46/0.069 |
| 1.2 | 0 | 0.05 | 1.20/0.052 | 1.14/0.056 |
| 1.4 | 0 | 0.05 | 1.36/0.062 | 1.30/0.063 |

Table 1: Data versus model predictions on images outside the training class. The first column of means and variances are from the experimental data, the second column from the model.

experimental data (subject mean and variance on the same image). The six free parameters of the model were described as the sum of third order polynomials of local $S, T$ and the noise levels. Gradient descent was used to train the model.

The model was trained and tested on three different subsets of the experimental data. When trained on data in which only $B$ varied, the model output accurately predicts unseen experimental data of the same type. When the data varied in $B$ and $\sigma_s$ or $\sigma_t$, the model outputs agree very well with subject data ($MRE \sim 5 - 8\%$). When trained on data where all three variables vary, the model fits the data reasonably well ($MRE \sim 8-13\%$). For a model of the first type, Figure 3 compares model predictions to data from within the same set, while Table 1 shows model outputs and subject responses for test examples from outside the training class.

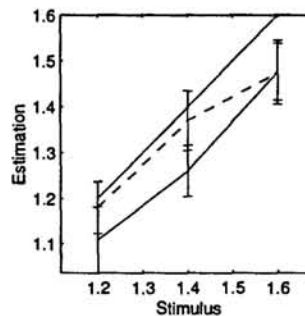

Figure 3: Model performance on data in which $\sigma_s = 0, \sigma_t = 0.10$. Upper line: perfect estimation. Lower line: experimental data. Dashed line: model prediction.

The model accounts for some important aspects of cue combination. Trained model parameters reveal that the texture prior is considerably weaker than the shading prior, and texture has a more reliable relationship with $B$. Consequently, at equal noise levels texture outweighs shading in the combination model. These factors account for the degree of underestimation found in each single-cue experiment, and the greater accuracy (i.e., enhancement) with combined-cues. Our studies also reveal a novel form of cue interaction: for some image patches, esp. at high curvature and noise levels, shading information becomes *harmful*, i.e., curvature estimation becomes less reliable when shading information is taken into account. Note that this differs from cue veto, in that texture does not veto shading.

Finally, the primary contribution of our model lies in its ability to predict the effect of continuous within-image variation in cue reliability on combination. Figure 4 shows how the estimation becomes more accurate and less variable with increas-

ing certainty in shading information. Standard cue combination models cannot produce similar behavior, as they do not estimate within-image cue reliabilities.

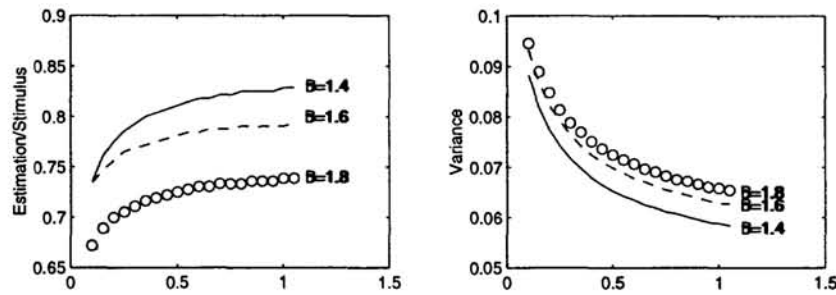

Figure 4: Mean (left) and variance (right) of model output as a function of $V_i^s$, for different values of $B$. Here $\sigma_s = 0.15, \sigma_t = 0$, all model parameters held constant.

## 5   CONCLUSION

We have proposed a hierarchical generative model to study cue combination. Inferring parameters from images is achieved by inverting this model. Inference produces probability distributions at each level: a set of local distributions, separately representing each cue, are combined to form a distribution over a relevant scene variable. The model naturally handles variations in cue reliability, which depend both on spatially local image context and general cue characteristics. This form of representation, incorporating image-specific cue utilities, makes this model more powerful than standard combination models. The model provides a good fit to our psychophysics results on shading and texture combination and an account for several aspects of cue combination; it also provides predictions for how varying noise levels, both within and across images, will effect combination.

We are extending this work in a number of directions. We are conducting experiments to obtain local shape estimates from subjects. We are considering better ways to extract local representations and distributions over them directly from an image, and methods of handling natural outliers such as shadows and occlusion.

## References

[1] Horn, B. K. P. (1977). Understanding image intensities. *AI 8*, 201-231.

[2] Jacobs, R. A. & Fine I. (1999). Experience-dependent integration of texture and motion cues to depth. *Vis. Res.*, 39, 4062-4075.

[3] Johnston, E. B., Cumming, B. G., & Landy, M. S. (1994). Integration of depth modules: Stereopsis and texture. *Vis. Res. 34*, 2259-2275.

[4] Knill, D. C. (1998). Surface orientation from texture: ideal observers, generic observers and the information content of texture cues. *Vis. Res. 38*, 1655-1682.

[5] Knill, D. C., Kersten, D., & Mamassian P. (1996). Implications of a Bayesian formulation of visual information for processing for psychophysics. In *Perception as Bayesian Inference*, D. C. Knill and W. Richards (Eds.), 239-286, Cambridge Univ Press.

[6] Landy, M. S., Maloney, L. T., Johnston, E. B., & Young, M. J. (1995). Measurement and modeling of depth cue combination: In defense of weak fusion. *Vis. Res. 35*, 389-412.

[7] Malik, J. & Rosenholtz, R. (1997). Computing local surface orientation and shape from texture for curved surfaces. *IJCV 23*, 149-168.

[8] Pentland, A. (1984). Local shading analysis. *IEEE PAMI, 6*, 170-187.

[9] Young, M.J., Landy, M.S., & Maloney, L.T. (1993). A perturbation analysis of depth perception from combinations of texture and motion cues. *Vis. Res. 33*, 2685-2696.

[10] Yuille, A. & Bulthoff, H. H. (1996). Bayesian decision theory and psychophysics. In *Perception as Bayesian Inference*, D. C. Knill and W. Richards (Eds.), 123-161, Cambridge Univ Press.

[11] Zemel, R. S., Dayan, P., & Pouget, A. (1998). Probabilistic interpretation of population codes. *Neural Computation*, 403-430.
